# Selecting Receptive Fields in Deep Networks

**Adam Coates**
Department of Computer Science
Stanford University
Stanford, CA 94305
acoates@cs.stanford.edu

**Andrew Y. Ng**
Department of Computer Science
Stanford University
Stanford, CA 94305
ang@cs.stanford.edu

## Abstract

Recent deep learning and unsupervised feature learning systems that learn from unlabeled data have achieved high performance in benchmarks by using extremely large architectures with many features (hidden units) at each layer. Unfortunately, for such large architectures the number of parameters can grow quadratically in the width of the network, thus necessitating hand-coded "local receptive fields" that limit the number of connections from lower level features to higher ones (e.g., based on spatial locality). In this paper we propose a fast method to choose these connections that may be incorporated into a wide variety of unsupervised training methods. Specifically, we choose local receptive fields that group together those low-level features that are most similar to each other according to a pairwise similarity metric. This approach allows us to harness the advantages of local receptive fields (such as improved scalability, and reduced data requirements) when we do not know how to specify such receptive fields by hand or where our unsupervised training algorithm has no obvious generalization to a topographic setting. We produce results showing how this method allows us to use even simple unsupervised training algorithms to train successful multi-layered networks that achieve state-of-the-art results on CIFAR and STL datasets: 82.0% and 60.1% accuracy, respectively.

## 1 Introduction

Much recent research has focused on training deep, multi-layered networks of feature extractors applied to challenging visual tasks like object recognition. An important practical concern in building such networks is to specify how the features in each layer connect to the features in the layers beneath. Traditionally, the number of parameters in networks for visual tasks is reduced by restricting higher level units to receive inputs only from a "receptive field" of lower-level inputs. For instance, in the first layer of a network used for object recognition it is common to connect each feature extractor to a small rectangular area within a larger image instead of connecting every feature to the entire image [14, 15]. This trick dramatically reduces the number of parameters that must be trained and is a key element of several state-of-the-art systems [4, 19, 6]. In this paper, we propose a method to automatically choose such receptive fields in situations where we do not know how to specify them by hand—a situation that, as we will explain, is commonly encountered in deep networks.

There are now many results in the literature indicating that large networks with thousands of unique feature extractors are top competitors in applications and benchmarks (e.g., [4, 6, 9, 19]). A major obstacle to scaling up these representations further is the blowup in the number of network parameters: for $n$ input features, a complete representation with $n$ features requires a matrix of $n^2$ weights—one weight for every feature and input. This blowup leads to a number of practical problems: (i) it becomes difficult to represent, and even more difficult to update, the entire weight matrix during learning, (ii) feature extraction becomes extremely slow, and (iii) many algorithms and techniques (like whitening and local contrast normalization) are difficult to generalize to large,

unstructured input domains. As mentioned above, we can solve this problem by limiting the "fan in" to each feature by connecting each feature extractor to a small receptive field of inputs. In this work, we will propose a method that chooses these receptive fields automatically during unsupervised training of deep networks. The scheme can operate without prior knowledge of the underlying data and is applicable to virtually any unsupervised feature learning or pre-training pipeline. In our experiments, we will show that when this method is combined with a recently proposed learning system, we can construct highly scalable architectures that achieve accuracy on CIFAR-10 and STL datasets beyond the best previously published.

It may not be clear yet why it is necessary to have an automated way to choose receptive fields since, after all, it is already common practice to pick receptive fields simply based on prior knowledge. However, this type of solution is insufficient for large, deep representations. For instance, in local receptive field architectures for image data, we typically train a bank of linear filters that apply only to a small image patch. These filters are then convolved with the input image to yield the first layer of features. As an example, if we train 100 5-by-5 pixel filters and convolve them with a 32-by-32 pixel input, then we will get a 28-by-28-by-100 array of features. Each 2D grid of 28-by-28 feature responses for a single filter is frequently called a "map" [14, 4]. Though there are still spatial relationships amongst the feature values within each map, it is not clear how two features in *different* maps are related. Thus when we train a second layer of features we must typically resort to connecting each feature to every input map or to a random subset of maps [12, 4] (though we may still take advantage of the remaining spatial organization within each map). At even higher layers of deep networks, this problem becomes extreme: our array of responses will have very small spatial resolution (e.g., 1-by-1) yet will have a large number of maps and thus we can no longer make use of spatial receptive fields. This problem is exacerbated further when we try to use very large numbers of maps which are often necessary to achieve top performance [4, 5].

In this work we propose a way to address the problem of choosing receptive fields that is not only a flexible addition to unsupervised learning and pre-training pipelines, but that can scale up to the extremely large networks used in state-of-the-art systems. In our method we select local receptive fields that group together (pre-trained) lower-level features according to a pairwise similarity metric between features. Each receptive field is constructed using a greedy selection scheme so that it contains features that are similar according to the similarity metric. Depending on the choice of metric, we can cause our system to choose receptive fields that are similar to those that might be learned implicitly by popular learning algorithms like ICA [11]. Given the learned receptive fields (groups of features) we can subsequently apply an unsupervised learning method independently over each receptive field. Thus, this method frees us to use any unsupervised learning algorithm to train the weights of the next layer. Using our method in conjunction with the pipeline proposed by [6], we demonstrate the ability to train multi-layered networks using only vector quantization as our unsupervised learning module. All of our results are achieved without supervised fine-tuning (i.e., backpropagation), and thus rely heavily on the success of the unsupervised learning procedure. Nevertheless, we attain the best known performances on the CIFAR-10 and STL datasets.

We will now discuss some additional work related to our approach in Section 2. Details of our method are given in Section 3 followed by our experimental results in Section 4.

## 2 Related Work

While much work has focused on different representations for deep networks, an orthogonal line of work has investigated the effect of network structure on performance of these systems. Much of this line of inquiry has sought to identify the best choices of network parameters such as size, activation function, pooling method and so on [12, 5, 3, 16, 19]. Through these investigations a handful of key factors have been identified that strongly influence performance (such as the type of pooling, activation function, and number of features). These works, however, do not address the finer-grained questions of how to choose the internal structure of deep networks directly.

Other authors have tackled the problem of architecture selection more generally. One approach is to search for the best architecture. For instance, Saxe et al. [18] propose using randomly initialized networks (forgoing the expense of training) to search for a high-performing structure. Pinto et al. [17], on the other hand, use a screening procedure to choose from amongst large numbers of randomly composed networks, collecting the best performing networks.

More powerful modeling and optimization techniques have also been used for learning the structure of deep networks in-situ. For instance, Adams et al. [1] use a non-parametric Bayesian prior to jointly infer the depth and number of hidden units at each layer of a deep belief network during training. Zhang and Chan [21] use an L1 penalization scheme to zero out many of the connections in an otherwise bipartite structure. Unfortunately, these methods require optimizations that are as complex or expensive as the algorithms they augment, thus making it difficult to achieve computational gains from any architectural knowledge discovered by these systems.

In this work, the receptive fields will be built by analyzing the relationships between feature responses rather than relying on prior knowledge of their organization. A popular alternative solution is to impose topographic organization on the feature outputs during training. In general, these learning algorithms train a set of features (usually linear filters) such that features nearby in a pre-specified topography share certain characteristics. The Topographic ICA algorithm [10], for instance, uses a probabilistic model that implies that nearby features in the topography have correlated variances (i.e., energies). This statistical measure of similarity is motivated by empirical observations of neurons and has been used in other analytical models [20]. Similar methods can be obtained by imposing group sparsity constraints so that features within a group tend to be on or off at the same time [7, 8]. These methods have many advantages but require us to specify a topography first, then solve a large-scale optimization problem in order to organize our features according to the given topographic layout. This will typically involve many epochs of training and repeated feature evaluations in order to succeed. In this work, we perform this procedure in reverse: our features are pre-trained using whatever method we like, then we will extract a useful grouping of the features post-hoc. This approach has the advantage that it can be scaled to large distributed clusters and is very generic, allowing us to potentially use different types of grouping criteria and learning strategies in the future with few changes. In that respect, part of the novelty in our approach is to convert existing notions of topography and statistical dependence in deep networks into a highly scalable "wrapper method" that can be re-used with other algorithms.

## 3 Algorithm Details

In this section we will describe our approach to selecting the connections between high-level features and their lower-level inputs (i.e., how to "learn" the receptive field structure of the high-level features) from an arbitrary set of data based on a particular pairwise similarity metric: square-correlation of feature responses.[1] We will then explain how our method integrates with a typical learning pipeline and, in particular, how to couple our algorithm with the feature learning system proposed in [6], which we adopt since it has been shown previously to perform well on image recognition tasks.

In what follows, we assume that we are given a dataset $X$ of feature vectors $x^{(i)}, i \in \{1, \ldots, m\}$, with elements $x_j^{(i)}$. These vectors may be raw features (e.g., pixel values) but will usually be features generated by lower layers of a deep network.

### 3.1 Similarity of Features

In order to group features together, we must first define a similarity metric between features. Ideally, we should group together features that are closely related (e.g., because they respond to similar patterns or tend to appear together). By putting such features in the same receptive field, we allow their relationship to be modeled more finely by higher level learning algorithms. Meanwhile, it also makes sense to model seemingly independent subsets of features separately, and thus we would like such features to end up in different receptive fields.

A number of criteria might be used to quantify this type of relationship between features. One popular choice is "square correlation" of feature responses, which partly underpins the Topographic ICA [10] algorithm. The idea is that if our dataset $X$ consists of linearly uncorrelated features (as can be obtained by applying a whitening procedure), then a measure of the higher-order dependence between two features can be obtained by looking at the correlation of their energies (squared responses). In particular, if we have $\mathbb{E}[x] = 0$ and $\mathbb{E}[xx^\top] = I$, then we will define the similarity

between features $x_j$ and $x_k$ as the correlation between the squared responses:

$$S[x_j, x_k] = \text{corr}(x_j^2, x_k^2) = \mathbb{E}\left[x_j^2 x_k^2 - 1\right] / \sqrt{\mathbb{E}\left[x_j^4 - 1\right]\mathbb{E}\left[x_k^4 - 1\right]}.$$

This metric is easy to compute by first whitening our input dataset with ZCA[2] whitening [2], then computing the pairwise similarities between all of the features:

$$S_{j,k} \equiv S_X[x_j, x_k] \equiv \frac{\sum_i {x_j^{(i)}}^2 {x_k^{(i)}}^2 - 1}{\sqrt{\sum_i ({x_j^{(i)}}^4 - 1) \sum_i ({x_k^{(i)}}^4 - 1)}}. \tag{1}$$

This computation is completely practical for fewer than 5000 input features. For fewer than 10000 features it is feasible but somewhat arduous: we must not only hold a 10000x10000 matrix in memory but we must also whiten our 10000-feature dataset—requiring a singular value or eigenvalue decomposition. We will explain how this expense can be avoided in Section 3.3, after we describe our receptive field learning procedure.

## 3.2 Selecting Local Receptive Fields

We now assume that we have available to us the matrix of pairwise similarities between features $S_{j,k}$ computed as above. Our goal is to construct "receptive fields": sets of features $R_n$, $n = 1, \ldots, N$ whose responses will become the inputs to one or more higher-level features. We would like for each $R_n$ to contain pairs of features with large values of $S_{j,k}$. We might achieve this using various agglomerative or spectral clustering methods, but we have found that a simple greedy procedure works well: we choose one feature as a seed, and then group it with its nearest neighbors according to the similarities $S_{j,k}$. In detail, we first select $N$ rows, $j_1, \ldots, j_N$, of the matrix $S$ at random (corresponding to a random choice of features $x_{j_n}$ to be the seed of each group). We then construct a receptive field $R_n$ that contains the features $x_k$ corresponding to the top $T$ values of $S_{j_n,k}$. We typically use $T = 200$, though our results are not too sensitive to this parameter. Upon completion, we have $N$ (possibly overlapping) receptive fields $R_n$ that can be used during training of the next layer of features.

## 3.3 Approximate Similarity

Computing the similarity matrix $S_{j,k}$ using square correlation is practical for fairly large numbers of features using the obvious procedure given above. However, if we want to learn receptive fields over huge numbers of features (as arise, for instance, when we use hundreds or thousands of maps), we may often be unable to compute $S$ directly. For instance, as explained above, if we use square correlation as our similarity criterion then we must perform whitening over a large number of features.

Note, however, that the greedy grouping scheme we use requires only $N$ rows of the matrix. Thus, provided we can compute $S_{j,k}$ for a single pair of features, we can avoid storing the entire matrix $S$. To avoid performing the whitening step for all of the input features, we can instead perform pair-wise whitening between features. Specifically, to compute the squared correlation of $x_j$ and $x_k$, we whiten the $j$th and $k$th columns of $X$ together (independently of all other columns), then compute the square correlation between the whitened values $\hat{x}_j$ and $\hat{x}_k$. Though this procedure is not equivalent to performing full whitening, it appears to yield effective estimates for the squared correlation between two features in practice. For instance, for a given "seed", the receptive field chosen using this approximation typically overlaps with the "true" receptive field (computed with full whitening) by 70% or more. More importantly, our final results (Section 4) are unchanged compared to the exact procedure.

Compared to the "brute force" computation of the similarity matrix, the approximation described above is very fast and easy to distribute across a cluster of machines. Specifically, the 2x2 ZCA whitening transform for a pair of features can be computed analytically, and thus we can express the pair-wise square correlations analytically as a function of the original inputs without having to

numerically perform the whitening on all pairs of features. If we assume that all of the input features of $x^{(i)}$ are zero-mean and unit variance, then we have:

$$\hat{x}_j^{(i)} = \frac{1}{2}((\gamma_{jk} + \beta_{jk})x_j^{(i)} + (\gamma_{jk} - \beta_{jk})x_k^{(i)})$$

$$\hat{x}_k^{(i)} = \frac{1}{2}((\gamma_{jk} - \beta_{jk})x_j^{(i)} + (\gamma_{jk} + \beta_{jk})x_k^{(i)})$$

where $\beta_{jk} = (1 - \alpha_{jk})^{-1/2}$, $\gamma_{jk} = (1 + \alpha_{jk})^{-1/2}$ and $\alpha_{jk}$ is the correlation between $x_j$ and $x_k$. Substituting $\hat{x}^{(i)}$ for $x^{(i)}$ in Equation 1 and expanding yields an expression for the similarity $S_{j,k}$ in terms of the pair-wise moments of each feature (up to fourth order). We can typically implement these computations in a single pass over the dataset that accumulates the needed statistics and then selects the receptive fields based on the results. Many alternative methods (e.g., Topographic ICA) would require some form of distributed optimization algorithm to achieve a similar result, which requires many feed-forward and feed-back passes over the dataset. In contrast, the above method is typically less expensive than a single feed-forward pass (to compute the feature values $x^{(i)}$) and is thus very fast compared to other conceivable solutions.

## 3.4   Learning Architecture

We have adopted the architecture of [6], which has previously been applied with success to image recognition problems. In this section we will briefly review this system as it is used in conjunction with our receptive field learning approach, but it should be noted that our basic method is equally applicable to many other choices of processing pipeline and unsupervised learning method.

The architecture proposed by [6] works by constructing a feature representation of a small image patch (say, a 6-by-6 pixel region) and then extracting these features from many overlapping patches within a larger image (much like a convolutional neural net).

Let $X \in \mathbb{R}^{m \times 108}$ be a dataset composed of a large number of 3-channel (RGB), 6-by-6 pixel image patches extracted from random locations in unlabeled training images and let $x^{(i)} \in \mathbb{R}^{108}$ be the vector of RGB pixel values representing the $i$th patch. Then the system in [6] applies the following procedure to learn a new representation of an image patch:

1. Normalize each example $x^{(i)}$ by subtracting out the mean and dividing by the norm. Apply a ZCA whitening transform to $x^{(i)}$ to yield $\hat{x}^{(i)}$.

2. Apply an unsupervised learning algorithm (e.g., K-means or sparse coding) to obtain a (normalized) set of linear filters (a "dictionary"), $\mathcal{D}$.

3. Define a mapping from the whitened input vectors $\hat{x}^{(i)}$ to output features given the dictionary $\mathcal{D}$. We use a soft threshold function that computes each feature $f_j^{(i)}$ as $f_j^{(i)} = \max\{0, \mathcal{D}^{(j)\top}\hat{x}^{(i)} - t\}$ for a fixed threshold $t$.

The computed feature values for each example, $f^{(i)}$, become the new representation for the patch $x^{(i)}$. We can now apply the learned feature extractor produced by this method to a larger image, say, a 32-by-32 pixel RGB color image. This large image can be represented generally as a long vector with $32 \times 32 \times 3 = 3072$ elements. To compute its feature representation we simply extract features from every overlapping patch within the image (using a stride of 1 pixel between patches) and then concatenate all of the features into a single vector, yielding a (usually large) new representation of the entire image.

Clearly we can modify this procedure to use choices of receptive fields other than 6-by-6 patches of images. Concretely, given the 32-by-32 pixel image, we could break it up into arbitrary choices of overlapping sets $R_n$ where each $R_n$ includes a subset of the RGB values of the whole image. Then we apply the procedure outlined above to each set of features $R_n$ independently, followed by concatenating all of the extracted features. In general, if $X$ is now any training set (not necessarily image patches), we can define $X_{R_n}$ as the training set $X$ reduced to include only the features in one receptive field, $R_n$ (that is, we simply discard all of the columns of $X$ that do not correspond to features in $R_n$). We may then apply the feature learning and extraction methods above to each $X_{R_n}$ separately, just as we would for the hand-chosen patch receptive fields used in previous work.

### 3.5 Network Details

The above components, conceptually, allow us to lump together arbitrary types and quantities of data into our unlabeled training set and then automatically partition them into receptive fields in order to learn higher-level features. The automated receptive field selection can choose receptive fields that span multiple feature maps, but the receptive fields will often span only small spatial areas (since features extracted from locations far apart tend to appear nearly independent). Thus, we will also exploit spatial knowledge to enable us to use large numbers of maps rather than trying to treat the entire input as unstructured data. Note that this is mainly to reduce the expense of feature extraction and to allow us to use spatial pooling (which introduces some invariance between layers of features); the receptive field selection method itself can be applied to hundreds of thousands of inputs. We now detail the network structure used for our experiments that incorporates this structure.

First, there is little point in applying the receptive field learning method to the raw pixel layer. Thus, we use 6-by-6 pixel receptive fields with a stride (step) of 1 pixel between them for the first layer of features. If the first layer contains $K_1$ maps (i.e., $K_1$ filters), then a 32-by-32 pixel color image takes on a 27-by-27-by-$K_1$ representation after the first layer of (convolutional) feature extraction. Second, depending on the unsupervised learning module, it can be difficult to learn features that are invariant to image transformations like translation. This is handled traditionally by incorporating "pooling" layers [3, 14]. Here we use average pooling over adjacent, disjoint 3-by-3 spatial blocks. Thus, applied to the 27-by-27-by-$K_1$ representation from layer 1, this yields a 9-by-9-by-$K1$ pooled representation.

After extracting the 9-by-9-by-$K_1$ pooled representation from the first two layers, we apply our receptive field selection method. We could certainly apply the algorithm to the entire high-dimensional representation. As explained above, it is useful to retain spatial structure so that we can perform spatial pooling and convolutional feature extraction. Thus, rather than applying our algorithm to the entire input, we apply the receptive field learning to 2-by-2 spatial regions within the 9-by-9-by-$K_1$ pooled representation. Thus the receptive field learning algorithm must find receptive fields to cover $2 \times 2 \times K_1$ inputs. The next layer of feature learning then operates on each receptive field within the 2-by-2 spatial regions separately. This is similar to the structure commonly employed by prior work [4, 12], but here we are able to choose receptive fields that span several feature maps in a deliberate way while also exploiting knowledge of the spatial structure.

In our experiments we will benchmark our system on image recognition datasets using $K_1 = 1600$ first layer maps and $K_2 = 3200$ second layer maps learned from $N = 32$ receptive fields. When we use three layers, we apply an additional 2-by-2 average pooling stage to the layer 2 outputs (with stride of 1) and then train $K_3 = 3200$ third layer maps (again with $N = 32$ receptive fields). To construct a final feature representation for classification, the outputs of the first and second layers of trained features are average-pooled over quadrants as is done by [6]. Thus, our first layer of features result in $1600 \times 4 = 6400$ values in the final feature vector, and our second layer of features results in $3200 \times 4 = 12800$ values. When using a third layer, we use average pooling over the entire image to yield 3200 additional feature values. The features for all layers are then concatenated into a single long vector and used to train a linear classifier (L2-SVM).

## 4 Experimental Results

We have applied our method to several benchmark visual recognition problems: the CIFAR-10 and STL datasets. In addition to training on the full CIFAR training set, we also provide results of our method when we use only 400 training examples per class to compare with other single-layer results in [6].

The CIFAR-10 examples are all 32-by-32 pixel color images. For the STL dataset, we downsample the (96 pixel) images to 32 pixels. We use the pipeline detailed in Section 3.4, with vector quantization (VQ) as the unsupervised learning module to train up to 3 layers. For each set of experiments we provide test results for 1 to 3 layers of features, where the receptive fields for the 2nd and 3rd layers of features are learned using the method of Section 3.2 and square-correlation for the similarity metric.

For comparison, we also provide test results in each case using several alternative receptive field choices. In particular, we have also tested architectures where we use a single receptive field ($N = 1$)

where $R_1$ contains all of the inputs, and random receptive fields ($N = 32$) where $R_n$ is filled according to the same algorithm as in Section 3.2, but where the matrix $S$ is set to random values. The first method corresponds to the "completely connected", brute-force case described in the introduction, while the second is the "randomly connected" case. Note that in these cases we use the same spatial organization outlined in Section 3.5. For instance, the completely-connected layers are connected to all the maps within a 2-by-2 spatial window. Finally, we will also provide test results using a larger 1st layer representation ($K_1 = 4800$ maps) to verify that the performance gains we achieve are not merely the result of passing more projections of the data to the supervised classification stage.

## 4.1 CIFAR-10

### 4.1.1 Learned 2nd-layer Receptive Fields and Features

Before we look at classification results, we first inspect the learned features and their receptive fields from the second layer (i.e., the features that take the pooled first-layer responses as their input). Figure 1 shows two typical examples of receptive fields chosen by our method when using square-correlation as the similarity metric. In both of the examples, the receptive field incorporates filters with similar orientation tuning but varying phase, frequency and, sometimes, varying color. The position of the filters within each window indicates its location in the 2-by-2 region considered by the learning algorithm. As we might expect, the filters in each group are visibly similar to those placed together by topographic methods like TICA that use related criteria.

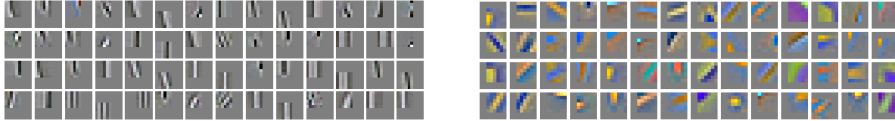

Figure 1: Two examples of receptive fields chosen from 2-by-2-by-1600 image representations. Each box shows the low-level filter and its position (ignoring pooling) in the 2-by-2 area considered by the algorithm. Only the most strongly dependent features from the $T = 200$ total features are shown. (Best viewed in color.)

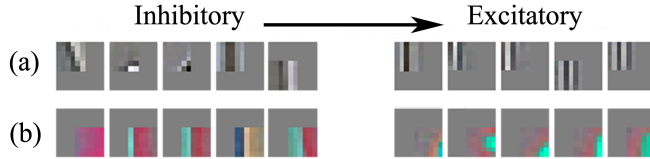

Figure 2: Most inhibitory (left) and excitatory (right) filters for two 2nd-layer features. (Best viewed in color.)

We also visualize some of the higher-level features constructed by the vector quantization algorithm when applied to these two receptive fields. The filters obtained from VQ assign weights to each of the lower level features in the receptive field. Those with a high positive weight are "excitatory" inputs (tending to lead to a high response when these input features are active) and those with a large negative weight are "inhibitory" inputs (tending to result in low filter responses). The 5 most inhibitory and excitatory inputs for two learned features are shown in Figure 2 (one from each receptive field in Figure 1). For instance, the two most excitatory filters of feature (a) tend to select for long, narrow vertical bars, inhibiting responses of wide bars.

### 4.1.2 Classification Results

We have tested our method on the task of image recognition using the CIFAR training and testing labels. Table 1 details our results using the full CIFAR dataset with various settings. We first note the comparison of our 2nd layer results with the alternative of a single large 1st layer using an equivalent number of maps (4800) and see that, indeed, our 2nd layer created with learned receptive fields performs better (81.2% vs. 80.6%). We also see that the random and single receptive field choices work poorly, barely matching the *smaller* single-layer network. This appears to confirm our belief that grouping together similar features is necessary to allow our unsupervised learning module (VQ) to identify useful higher-level structure in the data. Finally, with a third layer of features, we achieve the best result to date on the full CIFAR dataset with 82.0% accuracy.

Table 1: Results on CIFAR-10 (full)

| Architecture | Accuracy (%) |
|---|---|
| 1 Layer | 78.3% |
| 1 Layer (4800 maps) | 80.6% |
| 2 Layers (Single RF) | 77.4% |
| 2 Layers (Random RF) | 77.6% |
| 2 Layers (Learned RF) | 81.2% |
| 3 Layers (Learned RF) | **82.0%** |
| VQ (6000 maps) [6] | 81.5% |
| Conv. DBN [13] | 78.9% |
| Deep NN [4] | 80.49% |

Table 2: Results on CIFAR-10 (400 ex. per class)

| Architecture | Accuracy (%) |
|---|---|
| 1 Layer | 64.6% ($\pm$0.8%) |
| 1 Layer (4800 maps) | 63.7% ($\pm$0.7%) |
| 2 Layers (Single RF) | 65.8% ($\pm$0.3%) |
| 2 Layers (Random RF) | 65.8% ($\pm$0.9%) |
| 2 Layers (Learned RF) | 69.2% ($\pm$0.7%) |
| 3 Layers (Learned RF) | **70.7%** ($\pm$0.7%) |
| Sparse coding (1 layer) [6] | 66.4% ($\pm$0.8%) |
| VQ (1 layer) [6] | 64.4% ($\pm$1.0%) |

It is difficult to assess the strength of feature learning methods on the full CIFAR dataset because the performance may be attributed to the success of the supervised SVM training and not the unsupervised feature training. For this reason we have also performed classification using 400 labeled examples per class.[3] Our results for this scenario are in Table 2. There we see that our 2-layer architecture significantly outperforms our 1-layer system as well as the two 1-layer architectures developed in [6]. As with the full CIFAR dataset, we note that it was not possible to achieve equivalent performance by merely expanding the first layer or by using either of the alternative receptive field structures (which, again, make minimal gains over a single layer).

## 4.2 STL-10

Finally, we also tested our algorithm on the STL-10 dataset [5]. Compared to CIFAR, STL provides many fewer labeled training examples (allowing 100 labeled instances per class for each training fold). Instead of relying on labeled data, one tries to learn from the provided unlabeled dataset, which contains images from a distribution that is similar to the labeled set but broader. We used the same architecture for

Table 3: Classification Results on STL-10

| Architecture | Accuracy (%) |
|---|---|
| 1 Layer | 54.5% ($\pm$0.8%) |
| 1 Layer (4800 maps) | 53.8% ($\pm$1.6%) |
| 2 Layers (Single RF) | 55.0% ($\pm$0.8%) |
| 2 Layers (Random RF) | 54.4% ($\pm$1.2%) |
| 2 Layers (Learned RF) | 58.9% ($\pm$1.1%) |
| 3 Layers (Learned RF) | **60.1**% ($\pm$1.0%) |
| Sparse coding (1 layer) [6] | 59.0% ($\pm$0.8%) |
| VQ (1 layer) [6] | 54.9% ($\pm$0.4%) |

this dataset as for CIFAR, but rather than train our features each time on the labeled training fold (which is too small), we use 20000 examples taken from the unlabeled dataset. Our results are reported in Table 3.

Here we see increasing performance with higher levels of features once more, achieving state-of-the-art performance with our 3-layered model. This is especially notable since the higher level features have been trained purely from unlabeled data. We note, one more time, that none of the alternative architectures (which roughly represent common practice for training deep networks) makes significant gains over the single layer system.

## 5 Conclusions

We have proposed a method for selecting local receptive fields in deep networks. Inspired by the grouping behavior of topographic learning methods, our algorithm selects qualitatively similar groups of features directly using arbitrary choices of similarity metric, while also being compatible with any unsupervised learning algorithm we wish to use. For one metric in particular (square correlation) we have employed our algorithm to choose receptive fields within multi-layered networks that lead to successful image representations for classification while still using only vector quantization for unsupervised learning—a relatively simple by highly scalable learning module. Among our results, we have achieved the best published accuracy on CIFAR-10 and STL datasets. These performances are strengthened by the fact that they did not require the use of any supervised back-propagation algorithms. We expect that the method proposed here is a useful new tool for managing extremely large, higher-level feature representations where more traditional spatio-temporal local receptive fields are unhelpful or impossible to employ successfully.

## Footnotes

[1]Though we use this metric throughout, and propose some extensions, it can be replaced by many other choices such as the mutual information between two features.

[2]If $\mathbb{E}\left[xx^\top\right] = \Sigma = VDV^\top$, ZCA whitening uses the transform $P = VD^{-1/2}V^\top$ to compute the whitened vector $\hat{x}$ as $\hat{x} = Px$.

[3]Our networks are still trained unsupervised from the entire training set.

# References

[1] R. Adams, H. Wallach, and Z. Ghahramani. Learning the structure of deep sparse graphical models. In *International Conference on AI and Statistics*, 2010.

[2] A. Bell and T. J. Sejnowski. The 'independent components' of natural scenes are edge filters. *Vision Research*, 37, 1997.

[3] Y. Boureau, F. Bach, Y. LeCun, and J. Ponce. Learning mid-level features for recognition. In *Computer Vision and Pattern Recognition*, 2010.

[4] D. Ciresan, U. Meier, J. Masci, L. M. Gambardella, and J. Schmidhuber. High-performance neural networks for visual object classification. *Pre-print*, 2011. http://arxiv.org/abs/1102.0183.

[5] A. Coates, H. Lee, and A. Y. Ng. An analysis of single-layer networks in unsupervised feature learning. In *International Conference on AI and Statistics*, 2011.

[6] A. Coates and A. Y. Ng. The importance of encoding versus training with sparse coding and vector quantization. In *International Conference on Machine Learning*, 2011.

[7] P. Garrigues and B. Olshausen. Group sparse coding with a laplacian scale mixture prior. In *Advances in Neural Information Processing Systems*, 2010.

[8] K. Gregor and Y. LeCun. Emergence of complex-like cells in a temporal product network with local receptive fields, 2010.

[9] F. Huang and Y. LeCun. Large-scale learning with SVM and convolutional nets for generic object categorization. In *Computer Vision and Pattern Recognition*, 2006.

[10] A. Hyvarinen, P. Hoyer, and M. Inki. Topographic independent component analysis. *Neural Computation*, 13(7):1527–1558, 2001.

[11] A. Hyvarinen and E. Oja. Independent component analysis: algorithms and applications. *Neural networks*, 13(4-5):411–430, 2000.

[12] K. Jarrett, K. Kavukcuoglu, M. Ranzato, and Y. LeCun. What is the best multi-stage architecture for object recognition? In *International Conference on Computer Vision*, 2009.

[13] A. Krizhevsky. Convolutional Deep Belief Networks on CIFAR-10. Unpublished manuscript, 2010.

[14] Y. LeCun, F. Huang, and L. Bottou. Learning methods for generic object recognition with invariance to pose and lighting. In *Computer Vision and Pattern Recognition*, 2004.

[15] H. Lee, R. Grosse, R. Ranganath, and A. Y. Ng. Convolutional deep belief networks for scalable unsupervised learning of hierarchical representations. In *International Conference on Machine Learning*, 2009.

[16] V. Nair and G. E. Hinton. Rectified Linear Units Improve Restricted Boltzmann Machines. In *International Conference on Machine Learning*, 2010.

[17] N. Pinto, D. Doukhan, J. J. DiCarlo, and D. D. Cox. A high-throughput screening approach to discovering good forms of biologically inspired visual representation. *PLoS Comput Biol*, 2009.

[18] A. Saxe, P. Koh, Z. Chen, M. Bhand, B. Suresh, and A. Y. Ng. On random weights and unsupervised feature learning. In *International Conference on Machine Learning*, 2011.

[19] D. Scherer, A. Mller, and S. Behnke. Evaluation of pooling operations in convolutional architectures for object recognition. In *International Conference on Artificial Neural Networks*, 2010.

[20] E. Simoncelli and O. Schwartz. Modeling surround suppression in v1 neurons with a statistically derived normalization model. *Advances in Neural Information Processing Systems*, 1998.

[21] K. Zhang and L. Chan. Ica with sparse connections. *Intelligent Data Engineering and Automated Learning*, 2006.

